# Why The Brain Separates Face Recognition From Object Recognition

**Joel Z Leibo, Jim Mutch and Tomaso Poggio**
Department of Brain and Cognitive Sciences
Massachusetts Institute of Technology
Cambridge MA 02139
jzleibo@mit.edu, jmutch@mit.edu, tp@ai.mit.edu

## Abstract

Many studies have uncovered evidence that visual cortex contains specialized regions involved in processing faces but not other object classes. Recent electrophysiology studies of cells in several of these specialized regions revealed that at least some of these regions are organized in a hierarchical manner with viewpoint-specific cells projecting to downstream viewpoint-invariant identity-specific cells [1]. A separate computational line of reasoning leads to the claim that some transformations of visual inputs that preserve viewed object identity are class-specific. In particular, the 2D images evoked by a face undergoing a 3D rotation are not produced by the same image transformation (2D) that would produce the images evoked by an object of another class undergoing the same 3D rotation. However, within the class of faces, knowledge of the image transformation evoked by 3D rotation can be reliably transferred from previously viewed faces to help identify a novel face at a new viewpoint. We show, through computational simulations, that an architecture which applies this method of gaining invariance to class-specific transformations is effective when restricted to faces and fails spectacularly when applied to other object classes. We argue here that in order to accomplish viewpoint-invariant face identification from a single example view, visual cortex must separate the circuitry involved in discounting 3D rotations of faces from the generic circuitry involved in processing other objects. The resulting model of the ventral stream of visual cortex is consistent with the recent physiology results showing the hierarchical organization of the face processing network.

## 1 Introduction

There is increasing evidence that visual cortex contains discrete patches involved in processing faces but not other objects [2, 3, 4, 5, 6, 7]. Though progress has been made recently in characterizing the properties of these brain areas, the computational-level reason the brain adopts this modular architecture has remained unknown.

In this paper, we propose a new computational-level explanation for why visual cortex separates face processing from object processing. Our argument does not require us to claim that faces are automatically processed in ways that are inapplicable to objects (e.g. gaze detection, gender detection) or that cortical specialization for faces arises due to perceptual expertise [8], though the perspective that emerges from our model is consistent with both of these claims.

We show that the task of identifying individual faces in an optimally viewpoint invariant way from single training examples requires a separate neural circuitry specialized for faces. The crux of this identification problem involves discounting transformations of the target individual's appearance. *Generic* transformations e.g, translation, scaling and 2D in-plane rotation can be learned from any

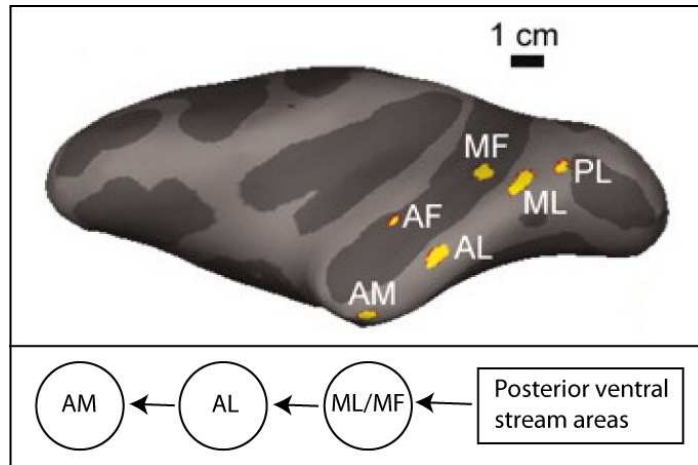

Figure 1: Layout of face-selective regions in macaque visual cortex, adapted from [1] with permission.

object class and usefully applied to any other class [9]. Other transformations which are *class-specific*, include changes in viewpoint and illumination. They depend on the object's 3D structure and material properties both of which vary between – but not within – certain object classes. Faces are the protoypical example of such a class where the individual objects are similar to each other. In this paper, we describe a method by which invariance to class-specific transformations can be encoded and used for within-class identification. The resulting model of visual cortex must separate the representations of different classes in order to achieve good performance.

This analysis is mainly computational but has implications for neuroscience and psychology. Section 2 of this paper describes the recently discovered hierarchical organization of the macaque face processing network [1]. Sections 3 and 4 describe an extension to an existing hierarchical model of object recognition to include invariances for class-specific transformations. The final section explains why the brain should have separate modules and relates the proposed computational model to physiology and neuroimaging evidence that the brain does indeed separate face recognition from object recognition.

## 2   The macaque face recognition hierarchy

In macaques, there are 6 discrete face-selective regions in the ventral visual pathway, one posterior lateral face patch (PL), two middle face patches (lateral- ML and fundus- MF), and three anterior face patches, the anterior fundus (AF), anterior lateral (AL), and anterior medial (AM) patches [5, 4]. At least some of these patches are organized into a feedforward hierarchy. Visual stimulation evokes a change in the local field potential $\sim 20$ ms earlier in ML/MF than in patch AM [1]. Consistent with a hierarchical organization involving information passing from ML/MF to AM via AL, electrical stimulation of ML elicited a response in AL and stimulation in AL elicited a response in AM [10].

The firing rates of cells in ML/MF are most strongly modulated by face viewpoint. Further along the hierarchy, in patch AM, cells are highly selective for individual faces but tolerate substantial changes in viewpoint [1].

The computational role of this recently discovered hierarchical organization is not yet established. In this paper, we argue that such a system – with view-tuned cells upstream from view-invariant identity-selective cells – is ideally suited to support face identification. In the subsequent sections, we present a model of the ventral stream that is consistent with a large body of experimental results[1] and additionally predicts the existence of discrete face-selective patches organized in this manner.

# 3 Hubel-Wiesel inspired hierarchical models of object recognition

At the end of the ventral visual pathway, cells in the most anterior parts of visual cortex respond selectively to highly complex stimuli and also invariantly over several degrees of visual angle. Hierarchical models inspired by Hubel and Wiesel's work, *H-W models*, seek to achieve similar selectivity and invariance properties by subjecting visual inputs to successive tuning and pooling operations [12, 13, 14, 15]. A major algorithmic claim made by these H-W models is that repeated application of this AND-like tuning operation is the source of the selective responses of cells at the end of the ventral stream. Likewise, repeated application of OR-like pooling operations yield invariant responses.

Hubel and Wiesel described complex cells as pooling the outputs of simple cells with the same optimal stimuli but receptive fields in different locations [16]. This pooling-over-position arrangement yields complex cells with larger receptive fields. That is, the operation transforms a position sensitive input to a (somewhat) translation invariant output. Similar pooling operations can also be employed to gain tolerance to other image transformations, including those induced by changes in viewpoint or illumination. Beyond V1, neurons can implement pooling just as they do within V1. Complex cells could pool over any transformation e.g., viewpoint, simply by connecting to (simple-like) cells that are selective for the appearance of the same feature at different viewpoints.

The specific H-W model which we extended in this paper is commonly known as HMAX [14, 17]; analogous extensions could be done for many related models. In this model, simple (S) cells compute a measure of their input's similarity to a stored optimal feature via a gaussian radial basis function or a normalized dot product. Complex (C) cells pool over S cells by computing the *max* response of all the S cells with which they are connected. These operations are typically repeated in a hierarchical manner, with the output of one C layer feeding into the next S layer and so on.

The max-pooling operation we employ can be viewed as an idealized mathematical description of the operation obtained by a system that has accurately associated template images across transformations. These associations could be acquired by a learning rule that connects input patterns that occur nearby in time to the same C unit. Numerous algorithms have been proposed to solve this invariance-learning problem through temporal association [18, 19, 20, 21, 22]. There is also psychophysical and physiological evidence that visual cortex employs a temporal association strategy[2] [23, 24, 25, 26, 27].

# 4 Invariance to class-specific transformations

H-W models can gain invariance to some transformations in a generic way. When the appearance of an input image under the transformation depends only on information available in a single example e.g., translation, scaling, and in-plane rotation, then the model's response to any image undergoing the transformation will remain constant no matter what templates were associated with one another to build the model. For example, a face can be encoded invariantly to translation as a vector of similarities to previously viewed template images of any other objects. The similarity "values" need not be high as long as they remain consistent across positions [9]. We refer to transformations with this property as generic, and note that they are the most common. Other transformations are class-specific, that is, they depend on information about the depicted object that is not available in a single image. For example, the 2D image evoked by an object undergoing a change in viewpoint depends on its 3D structure. Likewise, the images evoked by changes in illumination depend on the object's material properties. These class-specific properties can be learned from one or more exemplars of the class and applied to other objects in the class (see also [28, 29]). For this to work, the object class needs to consist of objects with similar 3D shape and material properties. Faces, as a class, are consistent enough in both 3D structure and material properties for this to work. Other, more diverse classes, such as "automobiles" are not.

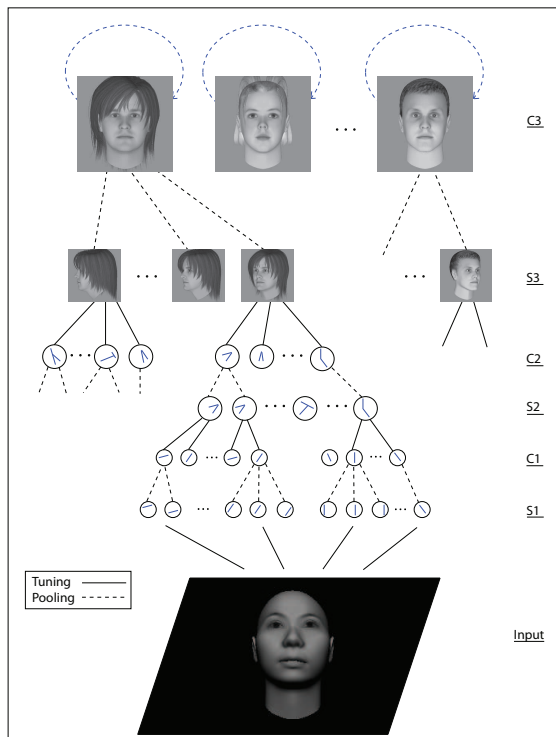

Figure 2: Illustration of an extension to the HMAX model to incorporate class-specific invariance to face viewpoint changes.

In our implementation of the HMAX model, the response of a C cell – associating templates $w$ at each position $t$ – is given by:

$$r_w(x) = \max_t \left( \exp\left( -\frac{1}{2\sigma} \sum_{j=1}^{n} (w_{t,j} - x_j)^2 \right) \right) \tag{1}$$

The same template $w_t$ is replicated at all positions, so this C response models the outcome of a temporal association learning process that associated the patterns evoked by a template at each position. This C response is invariant to translation. An analogous method can achieve viewpoint-tolerant responses. $r_w(x)$ is invariant to viewpoint changes of the input face $x$, as long as the 3D structure of the face depicted in the template images $w_t$ matches the 3D structure of the face depicted in $x$. Since all human faces have a relatively similar 3D structure, $r_w(x)$ will tolerate substantial viewpoint changes within the domain of faces. It follows that templates derived from a class of objects with the wrong 3D structure give rise to C cells that do not respond invariantly to 3D rotations.

Figures 3 and 4 show the performance of the extended HMAX model on viewpoint-invariant (fig3) and illumination-invariant (fig4) within-category identification tasks. Both of these are one-shot learning tasks. That is, a single view of a target object is encoded and a simple classifier (nearest neighbors) must rank test images depicting the same object as being more similar to the encoded target than to images of any other objects. Both targets and distractors were presented under varying viewpoints and illuminations. This task models the common situation of encountering a new face or object at one viewpoint and then being asked to recognize it again later from a different viewpoint.

The original HMAX model [14], represented here by the red curves (C2), shows a rapid decline in performance due to changes in viewpoint and illumination. In contrast, the C3 features of the extended HMAX model perform significantly better than C2. Additionally, the performance of the C3 features is not strongly affected by viewpoint and illumination changes (see the plots along the diagonal in figures 3I and 4I).

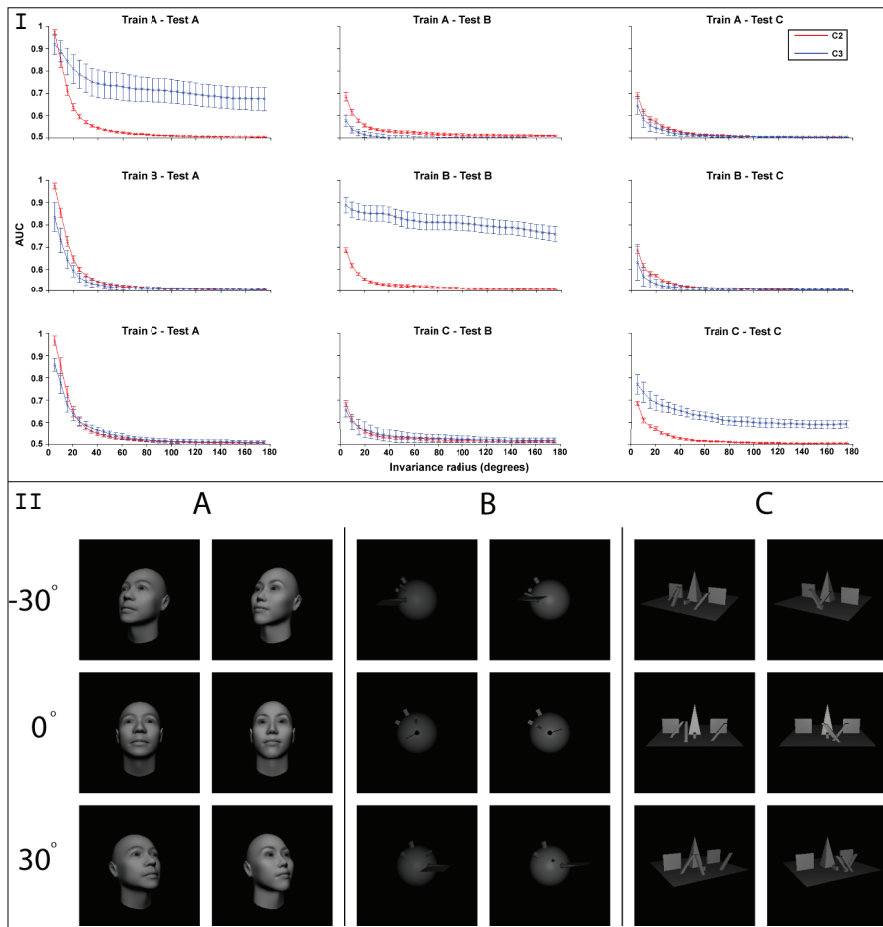

Figure 3: Viewpoint invariance. Bottom panel (II): Example images from three classes of stimuli. Class A consists of faces produced using FaceGen (Singular Inversions). Class B is a set of synthetic objects produced using Blender (Stichting Blender Foundation). Each object in this class has a central spike protruding from a sphere and two bumps always in the same location on top of the sphere. Individual objects differ from one another by the direction in which another protusion comes off of the central spike and the location/direction of an additional protrusion. Class C is another set of synthetic objects produced using Blender. Each object in this class has a central pyramid on a flat plane and two walls on either side. Individual objects differ in the location and slant of three additional bumps. For both faces and the synthetic classes, there is very little information to disambiguate individuals from views of the backs of the objects. Top panel (I): Each column shows the results of testing the model's viewpoint-invariant recognition performance on a different class of stimuli (A,B or C). The S3/C3 templates were obtained from objects in class A in the top row, class B in the middle row and class C in the bottom row. The abscissa of each plot shows the maximum invariance range (maximum deviation from the frontal view in either direction) over which targets and distractors were presented. The ordinate shows the AUC obtained for the task of recognizing an individual novel object despite changes in viewpoint. The model was never tested using the same images that were used to produce S3/C3 templates. A simple correlation-based nearest-neighbor classifier must rank all images of the same object at different viewpoints as being more similar to the frontal view than other objects. The red curves show the resulting AUC when the input to the classifier consists of C2 responses and the blue curves show the AUC obtained when the classifier's input is the C3 responses only. Simulation details: These simulations used 2000 translation and scaling invariant C2 units tuned to patches of natural images. The choice of natural image patches for S2/C2 templates had very little effect on the final results. Error bars (+/- one standard deviation) show the results of cross validation by randomly choosing a set of example images to use for producing S3/C3 templates and testing on the rest of the images. The above simulations used 710 S3 units (10 exemplar objects and 71 views) and 10 C3 units.

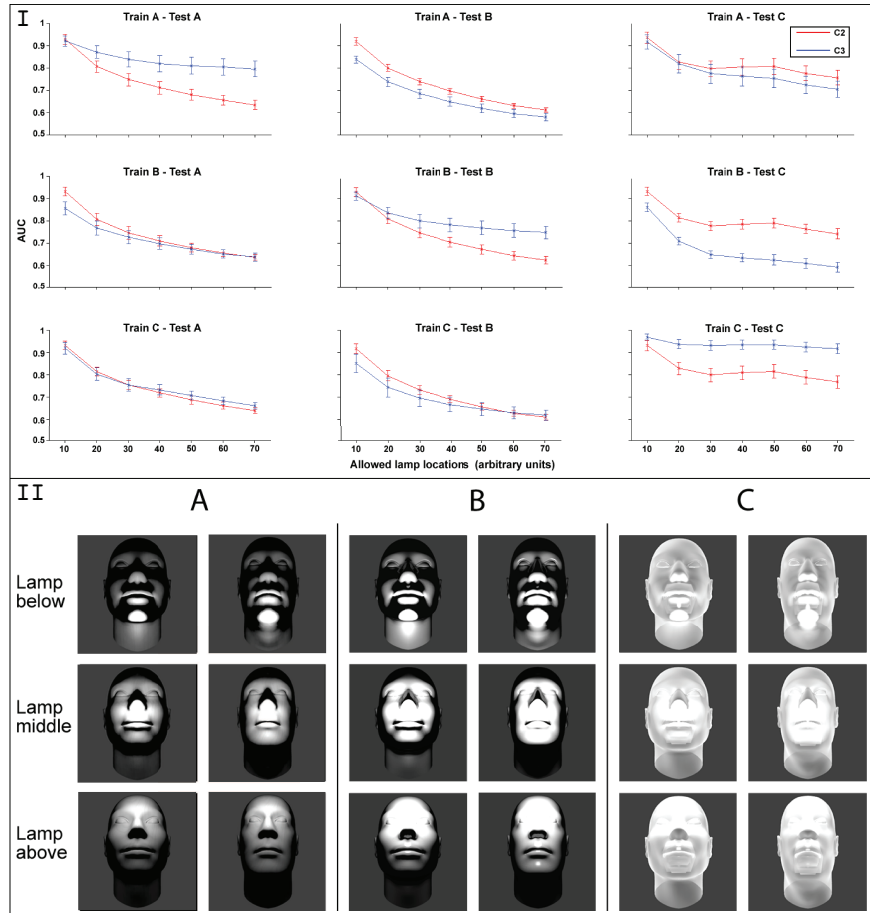

Figure 4: Illumination invariance. Same organization as in figure 3. Bottom panel (II): Example images from three classes of stimuli. Each class consists of faces with different light reflectance properties, modeling different materials. Class A was opaque and non-reflective like wood. Class B was opaque but highly reflective like a shiny metal. Class C was translucent like glass. Each image shows a face's appearance corresponding to a different location of the source of illumination (the lamp). The face models were produced using FaceGen and modifed with Blender. Top panel (I): Columns show the results of testing illumination-invariant recognition performance on class A (left), B (middle) and C (right). S3/C3 templates were obtained from objects in class A (top row), B (middle row), and C (bottom row). The model was never tested using the same images that were used to produce S3/C3 templates. As in figure 3, the abscissa of each plot shows the maximum invariance range (maximum distance the light could move in either direction away from a neutral position where the lamp is even with the middle of the head) over which targets and distractors were presented. The ordinate shows the AUC obtained for the task of recognizing an individual novel object despite changes in illumination. A correlation-based nearest-neighbor "classifier" must rank all images of the same object under each illumination condition as being more similar to the neutral view than other objects. The red curves show the resulting AUC when the input to the classifier consists of C2 responses and the blue curves show the AUC obtained when the classifier's input is the C3 responses only. Simulation details: These simulations used 80 translation and scaling invariant C2 units tuned to patches of natural images. The choice of natural image patches for S2/C2 templates had very little effect on the final results. Error bars (+/- one standard deviation) show the results of cross validation by randomly choosing a set of example images to use for producing S3/C3 templates and testing on the rest of the images. The above simulations used 1200 S3 units (80 exemplar faces and 15 illumination conditions) and 80 C3 units.

The C3 features are class-specific. Good performance on within-category identification is obtained using templates derived from the same category (plots along the diagonal in figures 3I and 4I). When C3 features from the wrong category are used in this way, performance suffers (off-diagonal plots). In all these cases, the C2 features which encode nothing specifically useful for taking into account the relevant transformation perform as well as or better than C3 features derived from objects of the wrong class. It follows that if the brain is using an algorithm of this sort (an H-W architecture) to accomplish within-category identification, then it must separate the circuitry that produces invariance for the transformations that objects of one class undergo from the circuitry producing invariance to the transformations that other classes undergo.

## 5   Conclusion

Everyday visual tasks require reasonably good invariance to non-generic transformations like changes in viewpoint and illumination[3]. We showed that a broad class of ventral stream models that is well-supported by physiology data (H-W models) require class-specific modules in order to accomplish these tasks.

The recently-discovered macaque face-processing hierarchy bears a strong resemblance to the architecture of our extended HMAX model. The responses of cells in an early part of the hierarchy (patches ML and MF) are strongly dependent on viewpoint, while the cells in a downstream area (patch AM) tolerate large changes in viewpoint. Identifying the S3 layer of our extended HMAX model with the ML/MF cells and the C3 layer with the AM cells is an intruiging possibility. Another mapping from the model to the physiology could be to identify the outputs of simple classifiers operating on C2, S3 or C3 layers with the responses of cells in ML/MF and AM.

Fundamentally, the 3D rotation of an object class with one 3D structure e.g., faces, is not the same as the 3D rotation of another class of objects with a different 3D structure. Generic circuitry cannot take into account both transformations at once. The same argument applies to all other non-generic transformations as well. Since the brain must take these transformations into account in interpreting the visual world, it follows that visual cortex must have a modular architecture. Object classes that are important enough to require invariance to these transformations of novel exemplars must be encoded by dedicated circuitry. Faces are clearly a sufficiently important category of objects to warrant this dedication of resources. Analogous arguments apply to a few other categories; human bodies all have a similar 3D structure and also need to be seen and recognized under a variety of viewpoint and illumination conditions, likewise, reading is an important enough activity that it makes sense to encode the visual transformations that words and letters undergo with dedicated circuitry (changes in font, viewing angle, etc). We do not think it is coincidental that, just as for faces, brain areas which are thought to be specialized for visual processing of the human body (the extrastriate body area [32]) and reading (the visual word form area [33, 34]) are consistently found in human fMRI experiments.

We have argued in favor of visual cortex implenting a modularity of *content* rather than *process*. The computations performed in each dedicated processing region can remain quite similar to the computations performed in other regions. Indeed, the connectivity within each region can be wired up in the same way, through temporal association. The only difference across areas is the object class (and the transformations) being encoded. In this view, visual cortex must be modular in order to succeed in the tasks with which it is faced.

**Acknowledgments**

This report describes research done at the Center for Biological & Computational Learning, which is in the McGovern Institute for Brain Research at MIT, as well as in the Dept. of Brain & Cognitive

Sciences, and which is affiliated with the Computer Sciences & Artificial Intelligence Laboratory (CSAIL). This research was sponsored by grants from DARPA (IPTO and DSO), National Science Foundation (NSF-0640097, NSF-0827427), AFSOR-THRL (FA8650-05-C-7262). Additional support was provided by: Adobe, Honda Research Institute USA, King Abdullah University Science and Technology grant to B. DeVore, NEC, Sony and especially by the Eugene McDermott Foundation.

## Footnotes

[1]See [11] for a review.

[2]These temporal association algorithms and the evidence for their employment by visual cortex are interesting in their own right. In this paper we sidestep the issue of how visual cortex associates similar features under different transformations in order to focus on the implications of having the representation that results from applying these learning rules.

[3]It is sometimes claimed that human vision is not viewpoint invariant [30]. It is certainly true that performance on psychophysical tasks requiring viewpoint invariance is worse than on tasks requiring translation invariance. This is fully consistent with our model. The 3D structure of faces does not vary wildly within the class, but there is certainly still some significant variation. It is this variability in 3D structure within the class that is the source of the model's imperfect performance. Many psychophysical experiments on viewpoint invariance were performed with synthetic "paperclip" objects defined entirely by their 3D structure. Our model predicts particularly weak performance on viewpoint-tolerance tasks with these stimuli and that is precisely what is observed [31].

# References

[1] W. Freiwald and D. Tsao, "Functional Compartmentalization and Viewpoint Generalization Within the Macaque Face-Processing System," *Science*, vol. 330, no. 6005, p. 845, 2010.

[2] N. Kanwisher, J. McDermott, and M. Chun, "The fusiform face area: a module in human extrastriate cortex specialized for face perception," *The Journal of Neuroscience*, vol. 17, no. 11, p. 4302, 1997.

[3] K. Grill-Spector, N. Knouf, and N. Kanwisher, "The fusiform face area subserves face perception, not generic within-category identification," *Nature Neuroscience*, vol. 7, no. 5, pp. 555–562, 2004.

[4] D. Tsao, W. Freiwald, R. Tootell, and M. Livingstone, "A cortical region consisting entirely of face-selective cells," *Science*, vol. 311, no. 5761, p. 670, 2006.

[5] D. Tsao, W. Freiwald, T. Knutsen, J. Mandeville, and R. Tootell, "Faces and objects in macaque cerebral cortex," *Nature Neuroscience*, vol. 6, no. 9, pp. 989–995, 2003.

[6] R. Rajimehr, J. Young, and R. Tootell, "An anterior temporal face patch in human cortex, predicted by macaque maps," *Proceedings of the National Academy of Sciences*, vol. 106, no. 6, p. 1995, 2009.

[7] S. Ku, A. Tolias, N. Logothetis, and J. Goense, "fMRI of the Face-Processing Network in the Ventral Temporal Lobe of Awake and Anesthetized Macaques," *Neuron*, vol. 70, no. 2, pp. 352–362, 2011.

[8] M. Tarr and I. Gauthier, "FFA: a flexible fusiform area for subordinate-level visual processing automatized by expertise," *Nature Neuroscience*, vol. 3, pp. 764–770, 2000.

[9] J. Z. Leibo, J. Mutch, L. Rosasco, S. Ullman, and T. Poggio, "Learning Generic Invariances in Object Recognition: Translation and Scale," *MIT-CSAIL-TR-2010-061, CBCL-294*, 2010.

[10] S. Moeller, W. Freiwald, and D. Tsao, "Patches with links: a unified system for processing faces in the macaque temporal lobe," *Science*, vol. 320, no. 5881, p. 1355, 2008.

[11] T. Serre, M. Kouh, C. Cadieu, U. Knoblich, G. Kreiman, and T. Poggio, "A theory of object recognition: computations and circuits in the feedforward path of the ventral stream in primate visual cortex," *CBCL Paper #259/AI Memo #2005-036*, 2005.

[12] K. Fukushima, "Neocognitron: A self-organizing neural network model for a mechanism of pattern recognition unaffected by shift in position," *Biological Cybernetics*, vol. 36, pp. 193–202, Apr. 1980.

[13] M. Riesenhuber and T. Poggio, "Hierarchical models of object recognition in cortex," *Nature Neuroscience*, vol. 2, pp. 1019–1025, Nov. 1999.

[14] T. Serre, L. Wolf, S. Bileschi, M. Riesenhuber, and T. Poggio, "Robust Object Recognition with Cortex-Like Mechanisms," *IEEE Trans. Pattern Anal. Mach. Intell.*, vol. 29, no. 3, pp. 411–426, 2007.

[15] B. W. Mel, "SEEMORE: Combining Color, Shape, and Texture Histogramming in a Neurally Inspired Approach to Visual Object Recognition," *Neural Computation*, vol. 9, pp. 777–804, May 1997.

[16] D. Hubel and T. Wiesel, "Receptive fields, binocular interaction and functional architecture in the cat's visual cortex," *The Journal of Physiology*, vol. 160, no. 1, p. 106, 1962.

[17] J. Mutch and D. Lowe, "Object class recognition and localization using sparse features with limited receptive fields," *International Journal of Computer Vision*, vol. 80, no. 1, pp. 45–57, 2008.

[18] P. Földiák, "Learning invariance from transformation sequences," *Neural Computation*, vol. 3, no. 2, pp. 194–200, 1991.

[19] S. Stringer and E. Rolls, "Invariant object recognition in the visual system with novel views of 3D objects," *Neural Computation*, vol. 14, no. 11, pp. 2585–2596, 2002.

[20] L. Wiskott and T. Sejnowski, "Slow feature analysis: Unsupervised learning of invariances," *Neural computation*, vol. 14, no. 4, pp. 715–770, 2002.

[21] T. Masquelier, T. Serre, S. Thorpe, and T. Poggio, "Learning complex cell invariance from natural videos: A plausibility proof," *AI Technical Report #2007-060 CBCL Paper #269*, 2007.

[22] M. Spratling, "Learning viewpoint invariant perceptual representations from cluttered images," *IEEE Transactions on Pattern Analysis and Machine Intelligence*, vol. 27, no. 5, pp. 753–761, 2005.

[23] D. Cox, P. Meier, N. Oertelt, and J. J. DiCarlo, "'Breaking'position-invariant object recognition," *Nature Neuroscience*, vol. 8, no. 9, pp. 1145–1147, 2005.

[24] N. Li and J. J. DiCarlo, "Unsupervised natural experience rapidly alters invariant object representation in visual cortex.," *Science*, vol. 321, pp. 1502–7, Sept. 2008.

[25] N. Li and J. J. DiCarlo, "Unsupervised Natural Visual Experience Rapidly Reshapes Size-Invariant Object Representation in Inferior Temporal Cortex," *Neuron*, vol. 67, no. 6, pp. 1062–1075, 2010.

[26] G. Wallis and H. H. Bülthoff, "Effects of temporal association on recognition memory.," *Proceedings of the National Academy of Sciences of the United States of America*, vol. 98, pp. 4800–4, Apr. 2001.

[27] G. Wallis, B. Backus, M. Langer, G. Huebner, and H. Bülthoff, "Learning illumination-and orientation-invariant representations of objects through temporal association," *Journal of vision*, vol. 9, no. 7, 2009.

[28] T. Vetter, A. Hurlbert, and T. Poggio, "View-based models of 3D object recognition: invariance to imaging transformations," *Cerebral Cortex*, vol. 5, no. 3, p. 261, 1995.

[29] E. Bart and S. Ullman, "Class-based feature matching across unrestricted transformations," *Pattern Analysis and Machine Intelligence, IEEE Transactions on*, vol. 30, no. 9, pp. 1618–1631, 2008.

[30] H. Bülthoff and S. Edelman, "Psychophysical support for a two-dimensional view interpolation theory of object recognition," *Proceedings of the National Academy of Sciences*, vol. 89, no. 1, p. 60, 1992.

[31] N. Logothetis, J. Pauls, H. Bülthoff, and T. Poggio, "View-dependent object recognition by monkeys," *Current Biology*, vol. 4, no. 5, pp. 401–414, 1994.

[32] P. Downing and Y. Jiang, "A cortical area selective for visual processing of the human body," *Science*, vol. 293, no. 5539, p. 2470, 2001.

[33] L. Cohen, S. Dehaene, and L. Naccache, "The visual word form area," *Brain*, vol. 123, no. 2, p. 291, 2000.

[34] C. Baker, J. Liu, L. Wald, K. Kwong, T. Benner, and N. Kanwisher, "Visual word processing and experiential origins of functional selectivity in human extrastriate cortex," *Proceedings of the National Academy of Sciences*, vol. 104, no. 21, p. 9087, 2007.

